# Learning from queries for maximum information gain in imperfectly learnable problems

Peter Sollich   David Saad
Department of Physics, University of Edinburgh
Edinburgh EH9 3JZ, U.K.
P.Sollich@ed.ac.uk, D.Saad@ed.ac.uk

## Abstract

In supervised learning, learning from queries rather than from random examples can improve generalization performance significantly. We study the performance of query learning for problems where the student cannot learn the teacher perfectly, which occur frequently in practice. As a prototypical scenario of this kind, we consider a linear perceptron student learning a binary perceptron teacher. Two kinds of queries for maximum information gain, *i.e.*, minimum entropy, are investigated: Minimum *student space* entropy (MSSE) queries, which are appropriate if the teacher space is unknown, and minimum *teacher space* entropy (MTSE) queries, which can be used if the teacher space is assumed to be known, but a student of a simpler form has deliberately been chosen. We find that for MSSE queries, the structure of the student space determines the efficacy of query learning, whereas MTSE queries lead to a higher generalization error than random examples, due to a lack of feedback about the progress of the student in the way queries are selected.

## 1   INTRODUCTION

In systems that learn from examples, the traditional approach has been to study generalization from random examples, where each example is an input-output pair

with the input chosen randomly from some fixed distribution and the corresponding output provided by a teacher that one is trying to approximate. However, random examples contain less and less new information as learning proceeds. Therefore, generalization performance can be improved by learning from queries, *i.e.*, by choosing the input of each new training example such that it will be, together with its expected output, in some sense 'maximally useful'. The most widely used measure of 'usefulness' is the information gain, *i.e.*, the decrease in entropy of the post-training probability distributions in the parameter space of the student or the teacher. We shall call the resulting queries 'minimum (student or teacher space) entropy (MSSE/MTSE) queries'; their effect on generalization performance has recently been investigated for *perfectly learnable* problems, where student and teacher space are identical (Seung *et al.*, 1992, Freund *et al.*, 1993, Sollich, 1994), and was found to depend qualitatively on the structure of the teacher. For a linear perceptron, for example, one obtains a relative reduction in generalization error compared to learning from random examples which becomes insignificant as the number of training examples, $p$, tends to infinity. For a perceptron with binary output, on the other hand, minimum entropy queries result in a generalization error which decays exponentially as $p$ increases, a marked improvement over the much slower algebraic decay with $p$ in the case of random examples.

In practical situations, one almost always encounters *imperfectly learnable* problems, where the student can only approximate the teacher, but not learn it perfectly. Imperfectly learnable problems can arise for two reasons: Firstly, the teacher space (*i.e.*, the space of models generating the data) might be unknown. Because the teacher space entropy is then also unknown, MSSE (and not MTSE) queries have to be used for query learning. Secondly, the teacher space may be known, but a student of a simpler structure might have deliberately been chosen to facilitate or speed up training, for example. In this case, MTSE queries could be employed as an alternative to MSSE queries. The motivation for doing this would be strongest if, as in the learning scenario that we consider below, it is known from analyses of perfectly learnable problems that the structure of the teacher space allows more significant improvements in generalization performance from query learning than the structure of the student space.

With the above motivation in mind, we investigate in this paper the performance of both MSSE and MTSE queries for a prototypical imperfectly learnable problem, in which a linear perceptron student is trained on data generated by a binary perceptron teacher. Both student and teacher are specified by an $N$-dimensional weight vector with real components, and we will consider the thermodynamic limit $N \to \infty, p \to \infty, \alpha = p/N = \text{const}$. In Section 2 below we calculate the generalization error for learning from random examples. In Sections 3 and 4 we compare the result to MSSE and MTSE queries. Throughout, we only outline the necessary calculations; for details, we refer the reader to a forthcoming publication. We conclude in Section 5 with a summary and brief discussion of our results.

## 2   LEARNING FROM RANDOM EXAMPLES

We denote students and teachers by $\mathcal{N}$ (for 'Neural network') and $\mathcal{V}$ (for 'element of the Version space', see Section 4), respectively, and their corresponding weight

vectors by $\mathbf{w}_{\mathcal{N}}$ and $\mathbf{w}_{\mathcal{V}}$. For an input vector $\mathbf{x}$, the outputs of a given student and teacher are

$$y_{\mathcal{N}} = \tfrac{1}{\sqrt{N}}\,\mathbf{x}^T\mathbf{w}_{\mathcal{N}}, \quad y_{\mathcal{V}} = \mathrm{sgn}\left(\tfrac{1}{\sqrt{N}}\,\mathbf{x}^T\mathbf{w}_{\mathcal{V}}\right).$$

Assuming that inputs are drawn from a uniform distribution over the hypersphere $\mathbf{x}^2 = N$, and taking as our error measure the standard squared output difference $\tfrac{1}{2}(y_{\mathcal{N}} - y_{\mathcal{V}})^2$, the generalization error, *i.e.*, the average error between student $\mathcal{N}$ and teacher $\mathcal{V}$ when tested on random test inputs, is given by

$$\epsilon_{\mathrm{g}}(\mathcal{N}, \mathcal{V}) = \frac{1}{2}\left[Q_{\mathcal{N}} + 1 - 2\frac{R}{\sqrt{Q_{\mathcal{V}}}}\left(\frac{2}{\pi}\right)^{1/2}\right], \tag{1}$$

where we have set $R = \tfrac{1}{N}\mathbf{w}_{\mathcal{N}}^T\mathbf{w}_{\mathcal{V}}, Q_{\mathcal{N}} = \tfrac{1}{N}\mathbf{w}_{\mathcal{N}}^2, Q_{\mathcal{V}} = \tfrac{1}{N}\mathbf{w}_{\mathcal{V}}^2$.

As our training algorithm we take stochastic gradient descent on the training error $E_{\mathrm{t}}$, which for a training set $\Theta^{(p)} = \{(\mathbf{x}^\mu, y^\mu = y_{\mathcal{V}}(\mathbf{x}^\mu)), \mu = 1 \ldots p\}$ is $E_{\mathrm{t}} = \tfrac{1}{2}\sum_\mu (y^\mu - y_{\mathcal{N}}(\mathbf{x}^\mu))^2$. A weight decay term $\tfrac{1}{2}\lambda\mathbf{w}_{\mathcal{N}}^2$ is added for regularization, *i.e.*, to prevent overfitting. Stochastic gradient descent on the resulting energy function $E = E_{\mathrm{t}} + \tfrac{1}{2}\lambda\mathbf{w}_{\mathcal{N}}^2$ yields a Gibbs post-training distribution of students, $P(\mathbf{w}_{\mathcal{N}}|\Theta^{(p)}) \propto \exp(-E/T)$, where the training temperature $T$ measures the amount of stochasticity in the training algorithm. For the linear perceptron students considered here, this distribution is Gaussian, with covariance matrix $T\mathbf{M}_{\mathcal{N}}^{-1}$, where ($\mathbf{1}_N$ denotes the $N \times N$ identity matrix)

$$\mathbf{M}_{\mathcal{N}} = \lambda\mathbf{1}_N + \tfrac{1}{N}\sum_{\mu=1}^p \mathbf{x}^\mu(\mathbf{x}^\mu)^T.$$

Since the length of the teacher weight vector $\mathbf{w}_{\mathcal{V}}$ does not affect the teacher outputs, we assume a spherical prior on teacher space, $P(\mathbf{w}_{\mathcal{V}}) \propto \delta(\mathbf{w}_{\mathcal{V}}^2 - N)$, for which $Q_{\mathcal{V}} = 1$. Restricting attention to the limit of zero training temperature, it is straightforward to calculate from eq. (1) the average generalization error obtained by training on random examples

$$\epsilon_{\mathrm{g}} - \epsilon_{\mathrm{g,min}} = \frac{1}{\pi}\left[\lambda_{\mathrm{opt}}G + \lambda(\lambda_{\mathrm{opt}} - \lambda)\frac{\partial G}{\partial\lambda}\right], \tag{2}$$

with the function $G = \langle\tfrac{1}{N}\mathrm{tr}\,\mathbf{M}_{\mathcal{N}}^{-1}\rangle_{P(\{\mathbf{x}^\mu\})}$ given by (Krogh and Hertz, 1992)

$$G = \frac{1}{2\lambda}\left[1 - \alpha - \lambda + \sqrt{(1-\alpha-\lambda)^2 + 4\lambda}\right]. \tag{3}$$

In eq. (2) we have explicitly subtracted the minimum achievable generalization error, $\epsilon_{\mathrm{g,min}} = \tfrac{1}{2}(1-2/\pi)$, which is nonzero since a linear perceptron cannot approximate a binary perceptron perfectly. At finite $\alpha$, the generalization error is minimized when the weight decay is set to its optimal value $\lambda = \lambda_{\mathrm{opt}} = \pi/2 - 1$. Note that since both $G$ and $\partial G/\partial\lambda$ tend to zero as $\alpha \to \infty$, the generalization error for random examples approaches the minimum achievable generalization error in this limit.

## 3   MINIMUM STUDENT SPACE ENTROPY QUERIES

We now calculate the generalization performance resulting from MSSE queries. For the training algorithm introduced in the last section, the student space entropy (normalized by $N$) is given by

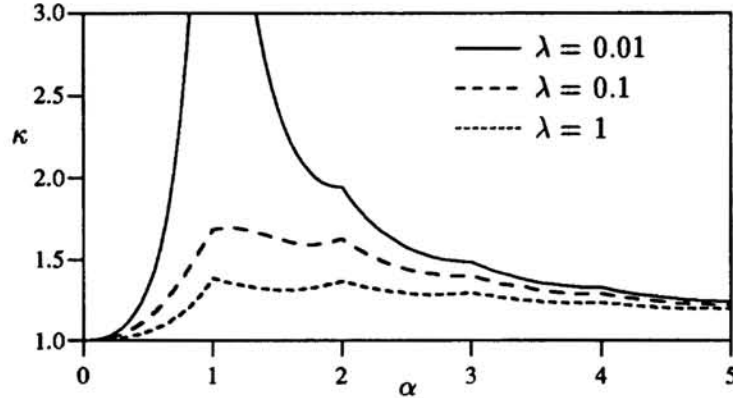

Figure 1: Relative improvement $\kappa$ in generalization error due to MSSE queries, for weight decay $\lambda = 0.01, 0.1, 1$.

$$S_\mathcal{N} = -\frac{1}{2N} \ln \det \mathbf{M}_\mathcal{N},$$

where we have omitted an unimportant constant which depends on the training temperature only. This entropy is minimized by choosing each new query along the direction corresponding to the minimal eigenvalue of the existing $\mathbf{M}_\mathcal{N}$ (Sollich, 1994). The expression for the resulting average generalization error is given by eq. (2) with $G$ replaced by its analogue for MSSE queries (Sollich, 1994)

$$G_Q = \frac{\Delta\alpha}{\lambda + [\alpha] + 1} + \frac{1 - \Delta\alpha}{\lambda + [\alpha]},$$

where $[\alpha]$ is the greatest integer less than or equal to $\alpha$ and $\Delta\alpha = \alpha - [\alpha]$. We define the improvement factor $\kappa$ as the ratio of the generalization error (with the minimum achievable generalization error subtracted as in eq. (2)) for random examples to that for MSSE queries. Figure 1 shows $\kappa(\alpha)$ for several values of the weight decay $\lambda$. Comparing with existing results (Sollich, 1994), we find that $\kappa$ is *exactly* the same as if our linear student were trying to approximate a linear teacher with additive noise of variance $\lambda_{\text{opt}}$ on the outputs. For large $\alpha$, one can show (Sollich, 1994) that $\kappa = 1 + 1/\alpha + O(1/\alpha^2)$ and hence the relative reduction in generalization error due to querying tends to zero as $\alpha \rightarrow \infty$. We investigate in the next section whether it is possible to improve generalization performance more significantly by using MTSE queries.

## 4   MINIMUM TEACHER SPACE ENTROPY QUERIES

We now consider the generalization performance achieved by MTSE queries. We remind readers that such queries could be used if the teacher space is known, but a student of a simpler functional form has deliberately chosen. The aim in using MTSE rather than MSSE queries would be to exploit the structure of the teacher space if this is known (for perfectly learnable problems) to make query learning very efficient compared to random examples.

For the case of noise free training data under consideration, the posterior probability distribution in teacher space given a certain training set is proportional to the prior

distribution on the version space (the set of all teachers that could have produced the training set without error) and zero everywhere else. From this the (normalized) teacher space entropy can be derived to be, up to an additive constant,

$$S_\nu = \frac{1}{N} \ln V(p),$$

where the version space volume $V(p)$ is given by ($\Theta(z) = 1$ for $z > 0$ and 0 otherwise)

$$V(p) = \int d\mathbf{w}_\nu \, P(\mathbf{w}_\nu) \prod_{\mu=1}^{p} \Theta\left(\frac{1}{\sqrt{N}} y^\mu \mathbf{w}_\nu^T \mathbf{x}^\mu\right).$$

It can easily be verified that this entropy is minimized[1] by choosing queries $\mathbf{x}$ which 'bisect' the existing version space, *i.e.*, for which the hyperplane perpendicular to $\mathbf{x}$ splits the version space into two equal halves (Seung *et al.*, 1992, Freund *et al.*, 1993). Such queries lead to an exponentially shrinking version space, $V(p) = 2^{-p}$, and hence a linear decrease of the entropy, $S_\nu = -\alpha \ln 2$. We consider instead queries which achieve qualitatively the same effect, but permit a much simpler analysis of the resulting student performance. They are similar to those studied in the context of a learnable problem by Watkin and Rau (1992), and are defined as follows. The $(p + 1)$th query is obtained by first picking a random teacher vector $\mathbf{w}_p$ from the version space defined by the existing $p$ training examples, and then picking the new training input $\mathbf{x}_{p+1}$ from the distribution of random inputs but under the constraint that $\mathbf{x}_{p+1}^T \mathbf{w}_p = 0$.

For the calculation of the student performance, *i.e.*, the average generalization error, achieved by the approximate MTSE queries described above, we use an approximation based on the following observation. As the number of training examples, $p$, increases, the teacher vectors $\mathbf{w}_p$ from the version space will align themselves with the true teacher $\mathbf{w}_\nu^0$; their components along the direction of $\mathbf{w}_\nu^0$ will increase, whereas their components perpendicular to $\mathbf{w}_\nu^0$ will decrease, varying widely across the $N-1$ dimensional hyperplane perpendicular to $\mathbf{w}_\nu^0$. Following Watkin and Rau (1992), we therefore assume that the only significant effect of choosing queries $\mathbf{x}_{p+1}$ with $\mathbf{x}_{p+1}^T \mathbf{w}_p = 0$ is on the distribution of the component of $\mathbf{x}_{p+1}$ along $\mathbf{w}_\nu^0$. Writing this component as $x_{p+1}^0 = \mathbf{x}_{p+1}^T \mathbf{w}_\nu^0 / |\mathbf{w}_\nu^0|$, its probability distribution can readily be shown to be

$$P(x_{p+1}^0) \propto \exp\left(-\tfrac{1}{2}(x_{p+1}^0/s_p)^2\right), \tag{4}$$

where $s_p$ is the sine of the angle between $\mathbf{w}_p$ and $\mathbf{w}_\nu^0$. For finite $N$, the value of $s_p$ is dependent on the $p$ previous training examples that define the existing version space and on the teacher vector $\mathbf{w}_p$ sampled randomly from this version space. In the thermodynamic limit, however, the variations of $s_p$ become vanishingly small and we can thus replace $s_p$ by its average value, which is a function of $p$ alone. In the thermodynamic limit, this average value becomes a continuous function of $\alpha = p/N$, the number of training examples per weight, which we denote simply by $s(\alpha)$. The calculation can then be split into two parts: First, the function $s(\alpha)$ is obtained from a calculation of the teacher space entropy using the replica method, generalizing the results of Györgi and Tishby (1990). The average generalization

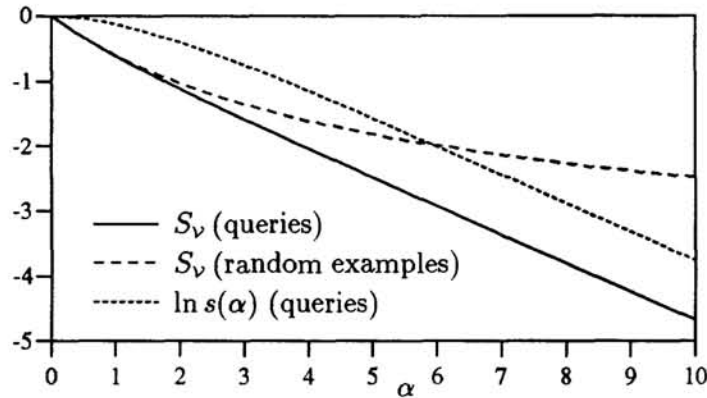

Figure 2: MTSE queries: Teacher space entropy, $S_\nu$ (with value for random examples plotted for comparison), and $\ln s$, the log of the sine of the angle between the true teacher and a random teacher from the version space.

error can then be calculated by using an extension of the response function method described in (Sollich, 1994b) or by another replica calculation (now in student space) as in (Dunmur and Wallace, 1993).

Figure 2 shows the effects of (approximate) MTSE queries in teacher space. For large $\alpha$ values, the teacher space entropy decreases linearly with $\alpha$, with gradient $c \approx 0.44$, whereas the entropy for random examples, also shown for comparison, decreases much more slowly (asymptotically like $-\ln \alpha$, see (Györgi and Tishby, 1990)). The linear $\alpha$-dependence of the entropy for queries corresponds to an average reduction of the version space volume with each new training example by a factor of $\exp(-c) \approx$ 0.64, which is reasonably close to the factor $\frac{1}{2}$ for proper bisection of the version space. This justifies our choice of analysing approximate MTSE queries rather than true MTSE queries, since the former achieve qualitatively the same results as the latter.

Before discussing the student performance achieved by (approximate) MTSE queries, we note from figure 2 that $\ln s(\alpha)$ decreases linearly with $\alpha$ for large $\alpha$, with the same gradient as the teacher space entropy. Hence $s(\alpha) \propto \exp(-c\alpha)$ for large $\alpha$, and MTSE queries force the average teacher from the version space to approach the true teacher exponentially quickly. It can easily be shown that if we were learning with a binary perceptron student, *i.e.*, if the problem were perfectly learnable, then this would result in an exponentially decaying generalization error, $\epsilon_g \propto \exp(-c\alpha)$. MTSE queries would thus lead to a marked improvement in generalization performance over random examples (for which $\epsilon_g \propto 1/\alpha$, see (Györgi and Tishby, 1990)). It is this significant benefit (in teacher space) of query learning that provides the motivation for using MTSE queries in imperfectly learnable problems such as the one considered here.

The results plotted in Figure 3 for the average generalization error achieved by the linear perceptron student show, however, that MTSE queries do not have the desired effect. Far from translating the benefits in teacher space into improvements in generalization performance for the linear student, they actually lead to a deterioration of generalization performance, *i.e.*, a larger generalization error than that

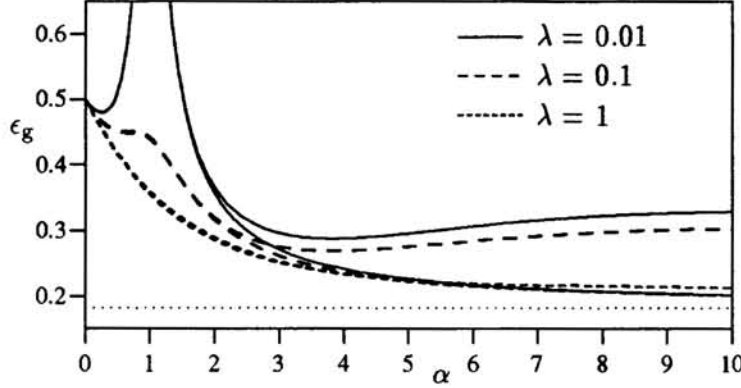

Figure 3: Generalization error for MTSE queries (higher curves of each pair) and random examples (lower curves), for weight decay $\lambda = 0.01, 0.1, 1$. The curves for random examples (which are virtually indistinguishable from one another already at $\alpha = 10$) converge to the minimum achievable generalization error $\epsilon_{g,min}$ (dotted line) as $\alpha \to \infty$.

obtained for random examples. Worse still, they 'mislead' the student to such an extent that the minimum achievable generalization error is not reached even for an infinite number of training examples, $\alpha \to \infty$. How does this happen? It can be verified that the angle between the student and teacher weight vectors tends to zero for $\alpha \to \infty$ as expected, while $Q_{\mathcal{N}}$, the normalized squared length of the student weight vector, approaches

$$Q_{\mathcal{N}}(\alpha \to \infty) = \frac{2}{\pi} \left( \frac{\bar{s}}{\lambda + \overline{s^2}} \right)^2 ,\qquad(5)$$

where $\bar{s} = \int_0^\infty d\alpha\, s(\alpha)$, $\overline{s^2} = \int_0^\infty d\alpha\, s^2(\alpha)$. Unless the weight decay parameter $\lambda$ happens to be equal to $\bar{s} - \overline{s^2}$, this is different from the optimal asymptotic value, which is $2/\pi$. This is the reason why in general the linear student does not reach the minimum possible generalization error even as $\alpha \to \infty$. The approach of $Q_{\mathcal{N}}$ to its non-optimal asymptotic value can cause an increase in the generalization error for large $\alpha$ and a corresponding minimum of the generalization error at some finite $\alpha$, as can be seen in the plots for $\lambda = 0.01$ and $0.1$ in Figure 3. For $\lambda = 0$, eq. (5) has the following intuitive interpretation: As $\alpha$ increases, the version space shrinks around the true teacher $\mathbf{w}_\nu^0$, and hence MTSE queries become 'more and more orthogonal' to $\mathbf{w}_\nu^0$. As a consequence, the distribution of training inputs along the direction of $\mathbf{w}_\nu^0$ is narrowed down progressively (compare eq. (4)). Trying to find a best fit to the teacher's binary output function over this narrower range of inputs, the linear student learns a function which is steeper than the best fit over the range of random inputs (which would give minimum generalization error). This corresponds to a suboptimally large length of the student weight vector in agreement with eq. (5): $Q_{\mathcal{N}}(\alpha \to \infty) > 2/\pi$ for $\lambda = 0$ because $\overline{s^2} < \bar{s}$.

Summarizing the results of this section, we have found that although MTSE queries are very beneficial in teacher space, they are entirely misleading for the linear student, to the extent that the student does not learn to approximate the teacher optimally even for an infinite number of training examples.

## 5   SUMMARY AND DISCUSSION

We have found in our study of an imperfectly learnable problem with a linear student and a binary teacher that queries for minimum student and teacher space entropy, respectively, have very different effects on generalization performance. Minimum student space entropy (MSSE) queries essentially have the same effect as for a linear student learning a noisy linear teacher, apart from a nonzero minimum value of the generalization error due to the unlearnability of the problem. Hence the structure of the student space is the dominating influence on the efficacy of query learning. Minimum teacher space entropy queries (MTSE), on the other hand, perform worse than random examples, leading to a higher generalization error even for an infinite number of training examples. With the benefit of hindsight, we note that this makes intuitive sense since the teacher space entropy, according to which MTSE queries are selected, contains no feedback about the progress of the student in learning the required generalization task, and thus MTSE queries cannot be guaranteed to have a positive effect.

Our results, then, are a mixture of good and bad news for query learning for maximum information gain in imperfectly learnable problems: The bad news is that MTSE queries, due to a lack of feedback information about student progress, are not enough to translate significant benefits in teacher space into similar improvements of student performance and may in fact yield worse performance than random examples. The good news is that for MSSE queries, we have found evidence that the structure of the student space is the key factor in determining the efficacy of query learning. If this result holds more generally, then statements about the benefits of query learning can be made on the basis of *how one is trying to learn* only, independently of *what one is trying to learn*—a result of great practical significance.

## Footnotes

[1]More precisely, what is minimized is the value of the entropy after a new training example $(\mathbf{x}, y)$ is added, averaged over the distribution of the unknown new training output $y$ given the new training input $\mathbf{x}$ and the existing training set; see Sollich (1994).

## References

A P Dunmur and D J Wallace (1993). Learning and generalization in a linear perceptron stochastically trained with noisy data. *J. Phys. A*, 26:5767–5779.

Y Freund, H S Seung, E Shamir, and N Tishby (1993). Information, prediction, and query by committee. In S J Hanson, J D Cowan, and C Lee Giles, editors, *NIPS 5*, pages 483–490, San Mateo, CA, Morgan Kaufmann.

G Györgi and N Tishby (1990). Statistical theory of learning a rule. In W Theumann and R Köberle, editors, *Neural Networks and Spin Glasses*, pages 3–36. Singapore, World Scientific.

A Krogh and J A Hertz (1992). Generalization in a linear perceptron in the presence of noise. *J. Phys. A*, 25:1135–1147.

P Sollich (1994). Query construction, entropy, and generalization in neural network models. *Phys. Rev. E*, 49:4637–4651.

P Sollich (1994b). Finite-size effects in learning and generalization in linear perceptrons. *J. Phys. A*, 27:7771–7784.

H S Seung, M Opper, and H Sompolinsky (1992). Query by committee. In *Proceedings of COLT '92*, pages 287–294, New York, ACM.

T L H Watkin and A Rau (1992). Selecting examples for perceptrons. *J. Phys. A*, 25:113–121.
